# Progressive Exploration-Conformal Learning for Sparsely Annotated Object Detection in Aerial Images

**Zihan Lu**[1], **Chenxu Wang**[1], **Chunyan Xu**[1,*], **Xiangwei Zheng**[2], **Zhen Cui**[1,*]

1. PCA Lab, Key Lab of Intelligent Perception and Systems for High-Dimensional
Information of Ministry of Education, School of Computer Science and
Engineering, Nanjing University of Science and Technology, Nanjing, China.
2. Shandong Normal University, Jinan, China.
{zihanlu, chenxuwang, cyx, zhen.cui}@njust.edu.cn, xwzhengcn@163.com

## Abstract

The ability to detect aerial objects with limited annotation is pivotal to the development of real-world aerial intelligence systems. In this work, we focus on a demanding but practical sparsely annotated object detection (SAOD) in aerial images, which encompasses a wider variety of aerial scenes with the same number of annotated objects. Although most existing SAOD methods rely on fixed thresholding to filter pseudo-labels for enhancing detector performance, adapting to aerial objects proves challenging due to the imbalanced probabilities/confidences associated with predicted aerial objects. To address this problem, we propose a novel Progressive Exploration-Conformal Learning (PECL) framework to address the SAOD task, which can adaptively perform the selection of high-quality pseudo-labels in aerial images. Specifically, the pseudo-label exploration can be formulated as a decision-making paradigm by adopting a conformal pseudo-label explorer and a multi-clue selection evaluator. The conformal pseudo-label explorer learns an adaptive policy by maximizing the cumulative reward, which can decide how to select these high-quality candidates by leveraging their essential characteristics and inter-instance contextual information. The multi-clue selection evaluator is designed to evaluate the explorer-guided pseudo-label selections by providing an instructive feedback for policy optimization. Finally, the explored pseudo-labels can be adopted to guide the optimization of aerial object detector in a closed-loop progressive fashion. Comprehensive evaluations on two public datasets demonstrate the superiority of our PECL when compared with other state-of-the-art methods in the sparsely annotated aerial object detection task. The code will be available at: https://github.com/SAOD-research/PECL.

## 1 Introduction

Recently, object detection has gained widespread attention, but the demand for a large amount of labeled data is time-consuming and labor-intensive. To address this challenge, semi-supervised object detection (SSOD) has been proposed to enhance detection performance by utilizing limited labeled samples and a large number of unlabeled samples. SSOD methods (14; 30; 23; 36) mainly focus on general images, and cannot consider the unique characteristics of objects in aerial images, such as dense arrangements, rich contextual relationships, and large complex scenes. As shown in Figure 1(a), objects in aerial images are more dense and similar compared to general images. For example, the average objects per image are 68.4 *vs* 7.7 in the DOTA (27) and COCO (13) datasets, respectively. Therefore, we address the sparsely annotated object detection (SAOD) task in aerial images, which

---

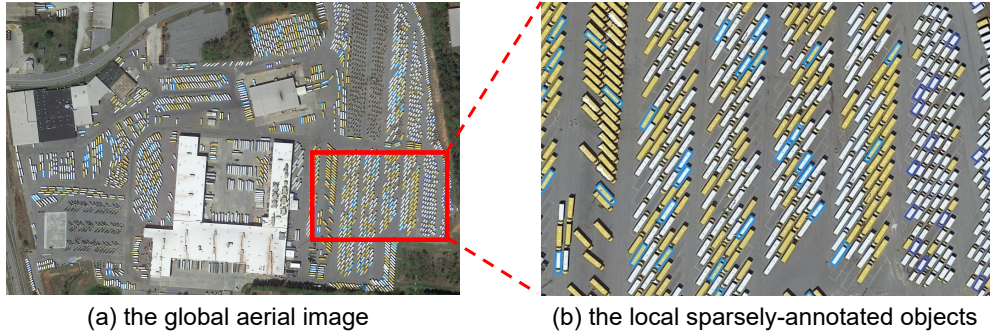

| (a) the global aerial image | (b) the local sparsely-annotated objects |

Figure 1: An example of sparsely annotated objects in an aerial image. Here a small part of objects are annotated, e.g., large vehicles with the light blue, and small vehicles with the dark blue.

labels a few part of objects in the training set. Compared with the conventional SSOD task, the SAOD can cover more diversity of aerial scenes even with the same number of annotated objects, as illustrated in Figure 1(b).

The SAOD task mitigates the need for expensive instance-level annotations, but also faces severe challenges of detector optimization, especially only with the limited and sparse annotation objects. In general, most existing SAOD approaches have attempted to mine more confidence supervised signals (e.g., pseudo-labels) from a substantial quantity of unlabeled aerial images. For example, Yoon et al. (32) combined an anchor-free object detector with an object tracker to generate dense annotations for training images. Co-mining (26) employed a Siamese network comprising two branches that mutually predict pseudo-label sets. Niitani et al. (18) utilized a part-aware sampling technique to handle instances within primary categories and used pseudo-labels to exclude un-annotated regions. However, existing SAOD methods typically focus on using fixed thresholds to filter pseudo-labels, lacking the ability to adaptively select high-quality pseudo-labels. Different from conventional objects, some aerial objects characterized by substantial dimensions and notable features (e.g., planes) exhibit higher predicted probabilities, whereas smaller aerial objects like vehicles are associated with comparatively lower predicted probabilities. Consequently, the discrepancy in probabilities and confidence levels observed in the predicted results poses a severe challenge in identifying pseudo-label instances by setting a rigorous threshold in the sparsely annotated aerial object detection process. Meanwhile, the conformal prediction methodology, as elucidated by Shafer et al. (2008) (20), emerges as a potent approach for quantifying uncertainty by harnessing confidence levels to effectively rectify the challenges stemming from imbalanced data.

In this work, we propose a Progressive Exploration-Conformable Learning (PECL) framework to improve the SAOD performance in aerial images, which can adaptively explore more confident pseudo-labels by considering the imbalance characteristics of different categories. Specifically, we firstly pre-train the detector with these sparsely annotated objects, and employ online clustering to generate class-wise knowledge to assist the pseudo-label exploration. The conformal decision problem of pseudo-label exploration is addressed with two essential components: the conformal pseudo-label explorer and multi-clue selection evaluator. The conformal pseudo-label explorer is responsible for selecting confident pseudo-labels by considering the intrinsic and conformal characteristics of the predicted candidates; while the multi-clue selection evaluator assesses the pseudo-label exploration and provides feedback to guide the optimization of the pseudo-label explorer. Finally, both the detector updating and the conformal pseudo-label exploration are integrated into a closed-loop, mutually reinforcing framework, promoting the SAOD task.

To summarize, the contributions of this work are threefold: i) We propose a progressive exploration-conformable learning framework that integrates the detector updating and the conformal pseudo-label exploration into an iteratively co-enhancing system. ii) We perform the pseudo-label exploration to mine more high-quality pseudo-labels by considering contextual information in large complex scenes, consisting of two modules: the conformal pseudo-label explorer and multi-clue selection evaluator. iii) Comprehensive evaluations on two public datasets, DOTA (27) and HRSC2016 (15), demonstrate the effectiveness of our PECL, which outperforms the baselines and state-of-the-art methods by a large margin in the sparsely annotated aerial object detection task.

## 2 Related Work

**Semi-/weakly-supervised/sparsely-annotated object detection:** Due to the extensive annotations requiring a significant investment of time and labor, SSOD and SAOD have attracted considerable attention in recent years. SSOD aims at the detector optimization by using limited labeled samples and a large number of unlabeled samples. For example, Pseudo labeling (10) has been widely used in semi-supervised learning, which utilizes pre-trained detector to generate labels for unlabeled data. Furthermore, some methods (2; 28; 11) combined consistency regularization with pseudo labeling. CSD (9) leveraged consistent predictions of horizontally flipped image pairs, STAC (21) enforced consistency constraints on weakly and strongly augmented image pairs. Unbiased Teacher (14) adopted the focal loss (12) to resolve pseudo-label bias caused by class imbalance in real labels. SOOD (7) designed rotation-aware adaptive weighting loss and global consistency loss for semi-supervised oriented object detection. Furthermore, considerable efforts have been made in weakly-supervised oriented bounding box detection. H2RBox (31) learned object center from horizontal box labels in weak-supervision, used scale and spatial constraints to estimate object dimensions in self-supervision. PointOBB (16) and Point2RBox (34) used Multiple Instance Learning and Knowledge Combination to learn rotated box regression from single point supervision. While SAOD labels part of instances in each image. Many researchers have explored methods to address this challenge. Niitani et al. (18) introduced part-aware sampling that leveraged the logical relationship between parts to guide the sampling process. Co-mining (26) involved a Siamese network with two branches that predicted pseudo-label sets for each other. Rambhatla et al. (19) treated SAOD as a semi-supervised problem focusing on regions, automatically identifying unlabeled foreground objects. Calibrated Teacher (25) calibrated the predicted confidence to match the actual accuracy, ensuring detectors at different training stages share similar confidence distributions and fixed threshold. In contrast, our work focuses on sparsely annotated aerial object detection by adaptively exploring contextual relationships between unlabeled instances in large complex scenes.

**Reinforcement/exploratory learning in computer vision:** Deep reinforcement learning has shown promising results in many decision-making domains. It has also been widely applied in the field of computer vision. Caicedo et al. (3) transformed the object detection into a Markov Decision Process to deform bounding boxes for accurate localization. Huang et al. (8) proposed an adaptive tracking method, which used reinforcement learning to select the number of network layers. DSN (37) treated video summarization as a sequence prediction problem, predicting frame probabilities and selecting frames based on the probability. Yu et al. (33) applied reinforcement learning for image restoration, employing a policy network to select suitable repair tools. Walid et al. (1) formulated the landmark localization in 3D medical images as a reinforcement learning problem and introduced actor-critic for localization tasks. Tian et al. (24) introduced a medical image segmentation method based on reinforcement learning, the reward showed the change between the predictions and the ground truth. IVADC-FDRL (17) empowered the agent to learn from real data by integrating detector for anomaly detection and Q-learning for anomaly classification. In this work, we build a progressive exploration-conformal learning process to adaptively select high-quality pseudo-labels, thereby enhancing the supervision signal for sparsely annotated aerial object detection.

## 3 The Proposed Method

### 3.1 Problem Definition

The sparsely annotated aerial object detection is to learn a robust aerial detector by employing these sparsely annotated instances and a large amount of unlabeled images/regions in the training set. Formally, we define $\mathcal{X} = \{X_i, O_i\}_{i=1}^{N}$ as the training set, where $N$ represents the total number of training images, $X_i$ and $O_i$ denote the $i$-th image sample and the corresponding sparsely annotated object set, respectively. For the $i$-th image, the annotated object set can be represented as $O_i = \{x_i^j, y_i^j, b_i^j\}_{j=1}^{N_{il}}$, where $x_i^j, y_i^j, b_i^j$ denote the image region, the class label, the bounding-box location of the $j$-th annotated object, and $N_{il}$ (i.e., $N_{il} \geq 0$) refers to the total number of labeled instances.

In the sparsely annotated aerial object detection task, how to mine more confident supervision (e.g., pseudo-labels) from these unlabeled images/regions is critical to promote the detector optimization. In general, most existing sparsely annotated object detection approaches (10; 26; 25) adopted a fixed threshold to select pseudo-labels, which can be used to provide more supervised signals. Unlike these typical objects, certain aerial objects characterized by their large, distinct features (e.g.,

planes) exhibit heightened predicted probabilities, whereas aerial objects with small dimensions (e.g., vehicles) tend to have lower predicted probabilities. Therefore, the imbalance in the predicted object probabilities could pose a challenge in pinpointing pseudo-label samples through a strict thresholding in the sparsely annotated aerial object detection problem. Inspired by the conformal prediction technique (20), which enables uncertainty quantification and utilizes confidence levels to tackle imbalanced data, we propose a novel Progressive Exploration-Conformable Learning (PECL) framework to address the SAOD task in aerial images.

## 3.2 PECL framework

Our proposed PECL framework can be built as a close-loop iterative learning process between the conformal pseudo-label exploration and the detector updating. Firstly, we can obtain a pre-trained aerial detector $\Theta$ by employing these sparsely annotated objects in the training set. To provide the confident guide for the subsequent process, we construct class-wise prototypes $\mathbb{P} = \left\{ p_{ck} \in \mathbb{R}^d \right\}_{c=1,k=1}^{C,K}$ with the fully connected layer features of each annotated instance as in (38). Here $C$ is the number of classes, $K$ is the number of prototypes per class, and $d$ is the feature dimension of the prototype. We can obtain the candidate set of each input sample $X_i$, i.e., $\{\tilde{x}_i^t\}_{t=1}^{N_{ic}}$, where $N_{ic}$ represents the number of candidates. We perform the conformal pseudo-label exploration to refine and identify these high-quality candidates as pseudo-labels, which can adaptively consider the distinct characteristics of different categories in the candidate set, especially in the aerial images. Here we develop the conformal pseudo-label explorer to facilitate the creation of an adaptive exploration policy, and the multi-clue selection evaluator to appraise the effectiveness of pseudo-label selection.

### 3.2.1 Conformal Pseudo-label Explorer

To address the imbalanced prediction probabilities across different categories in the aerial scenes, we specially design the conformal pseudo-label explorer to effectively probe these high-quality pseudo-labels. Specifically, the conformal pseudo-label explorer $\pi$ is a multi-layer perceptron composed of three fully connected layers. Its objective is to learn an adaptive pseudo-label exploration policy to determine whether to select the current candidate $\tilde{x}_i^t$. The conformal pseudo-label explorer $\pi$ takes the current exploratory characteristic $c_i^t$ as input, and obtains a two-dimensional selection probability distribution $\pi(c_i^t) \in \mathbb{R}^2$, representing the probabilities of taking different selections under the current characteristic. The selection decision $a_i^t$ is then obtained by sampling from this distribution, i.e., $a_i^t \sim \pi(c_i^t)$, where a decision value of 0 indicates not selecting the candidate, a value of 1 represents selecting the candidate as a pseudo-label. To assist the conformal pseudo-label explorer $\pi$ in performing the rational decision, the important point is how to design a comprehensive exploratory characteristic $c_i^t$ by considering multiple aspects of information. Here the characteristic $c_i^t = F_c\{O_i, \tilde{O}_i^t, \tilde{x}_i^t\}$ is formed by utilizing the characteristics of the sparsely annotated objects $O_i$, the selected pseudo-labels $\tilde{O}_i^t$, and the current candidate $\tilde{x}_i^t$ in the current $i$-th aerial image. The exploratory characteristic thus provides a rich and accurate scene description for the conformal pseudo-label explorer, facilitating more effective pseudo-label selection.

Taking the candidate $\tilde{x}_i^t$ as an example, we explain a detailed depiction of constructing the corresponding $c_i^t$ at time $t$. The characteristic of the current candidate $\tilde{x}_i^t$ can be represented as a feature vector, denoted as $[\mathcal{A}_{\mathbb{P}}(\tilde{x}_i^t), pro(\tilde{x}_i^t), f_{sim}(\tilde{x}_i^t)]$. $\mathcal{A}_{\mathbb{P}}(\tilde{x}_i^t)$ determines the precise level of confidence (20) for the candidate $\tilde{x}_i^t$ to quantify the uncertainty of the predicted probability. Here $\mathcal{A}_{\mathbb{P}}(\tilde{x}_i^t)$ can be expressed as a class-conditional probability where the non-conformity scores of all learned prototypes take on a value bigger than or equal to the non-conformity score of this candidate under the condition of the certain class $\tilde{y}_i^t$. Formally:

$$\mathcal{A}_{\mathbb{P}}(\tilde{x}_i^t) = Pro\left( \{NC(p_{ck})\}_{c=1,k=1}^{C,K} \geq NC(\tilde{x}_i^t)|\tilde{y}_i^t \right)$$
$$= \frac{\sum_{k=1}^{K} \mathbb{I}\left[ NC(\tilde{y}_i^t|p_{\tilde{y}_i^t k}) \geq NC(\tilde{y}_i^t|\tilde{x}_i^t) \right] + 1}{K + 1} \tag{1}$$

where $NC(\tilde{y}_i^t|\tilde{x}_i^t) = 1 - pro(\tilde{y}_i^t|\tilde{x}_i^t)$ represents the non-conformity score of the candidate $\tilde{x}_i^t$ with the class $\tilde{y}_i^t$ and $\mathbb{I}[\cdot]$ is the indicator function. A higher non-conformity score indicates a greater dissimilarity between the candidate and the reference class distribution. $pro(\tilde{x}_i^t)$ indicates the predicted probability of candidate $\tilde{x}_i^t$, inferred by the optimized detector. $f_{sim}(\tilde{x}_i^t)$ computes the

maximum cosine similarity between the feature of the candidate $\tilde{x}_i^t$ and the prototypes $\{p_{\tilde{y}_i^t k}\}_{k=1}^K$ of the predicted class label $\tilde{y}_i^t$. Based on the above, we can obtain current exploratory characteristic $c_i^t$ by concatenating the average feature of the sparsely annotated objects $O_i$ and the selected pseudo-labels $\tilde{O}_i^t$ with the feature of the current candidate $\tilde{x}_i^t$. The conformal exploratory characteristic enables the explorer to make effective pseudo-label selection and explore the unlabeled candidates efficiently, leading to the improved performance of the detector.

### 3.2.2 Multi-clue Selection Evaluator

To facilitate the conformal pseudo-label explorer $\pi$ in learning optimal pseudo-label exploration policy, we introduce a multi-clue selection evaluator $Q$ to assess the current exploratory characteristic $c_i^t$ and selection policy $a_i^t$, providing instructive feedback for policy optimization. In essence, the selection evaluator is also constructed as a multi-layer network with three fully connected layers. It takes the current characteristic $c_i^t$ and the decision policy $a_i^t$ as input and estimates the cumulative reward for the characteristic-decision pairs (i.e., $Q(c_i^t, a_i^t)$), signifying the effectiveness of the current policy. The cumulative reward takes into account the instant exploratory reward $r_i^t$ accumulated at each time step $t$ during the conformal pseudo-label decision, which measures how well the conformal pseudo-label explorer $\pi$ performs. A high cumulative reward indicates that the conformal pseudo-label explorer $\pi$ achieves positive candidate selection in decision-making process, and vice versa.

To accurately estimate the cumulative reward, the instant exploratory reward $r_i^t$ should reflect whether the current selection $a_i^t$ is appropriate or not. We first design a reward function $\psi(\tilde{x}_i^t)$ to evaluate the impact of selecting current candidate $\tilde{x}_i^t$ from the perspectives of information entropy and confidence margin, which is specifically represented as follows:

$$\psi(\tilde{x}_i^t) = \Delta H(\tilde{x}_i^t) + \xi \Delta U(\tilde{x}_i^t) \tag{2}$$

where $\xi$ is the weighting parameter. $\Delta H(\tilde{x}_i^t)$ represents the change in the information entropy of the current confident instances belonging to the same class $\tilde{y}_i^t$ when the current candidate $\tilde{x}_i^t$ is introduced into the collection of the sparsely annotated objects $O_i$ and the selected pseudo-labels $\tilde{O}_i^t$. It can be expressed as:

$$\begin{aligned} \Delta H(\tilde{x}_i^t) &= H(\tilde{\mu}_i^t) - H(\mu_i^t) \\ s.t. \ \tilde{\mu}_i^t &= \tfrac{1}{M+1}(\textstyle\sum_{m=1}^M pro(\bar{x}_i^m) + pro(\tilde{x}_i^t)) \\ \mu_i^t &= \tfrac{1}{M}\textstyle\sum_{m=1}^M pro(\bar{x}_i^m) \end{aligned} \tag{3}$$

where $H(\cdot)$ calculates the information entropy of probability distribution. $\bar{x}_i^m$ is from the set $\{O_i \cup \tilde{O}_i^t \cup \mathbb{P}\}_{\tilde{y}_i^t}$ related to the class $\tilde{y}_i^t$ and $M$ denotes its total number. The candidate with a high uncertainty will increase the overall entropy. $\Delta U(\tilde{x}_i^t)$ indicates the change in the confidence margin of the current confident instances belonging to the predicted class $\tilde{y}_i^t$ when the current candidate $\tilde{x}_i^t$ is selected. It can be formulated as:

$$\Delta U(\tilde{x}_i^t) = U(\mu_i^t) - U(\tilde{\mu}_i^t) \tag{4}$$

where $U(\cdot)$ denotes the confidence margin, calculating the difference between the highest value and the second highest value in the prediction probability. A larger margin indicates that the candidate is more confident, leading to a positive promotion in the overall confidence margin.

By considering the reward function $\psi(\tilde{x}_i^t)$ of the current candidate and the current selection $a_i^t$ taken by the conformal pseudo-label explorer, we ultimately design the instant exploratory reward $r_i^t$ as follows:

$$r_i^t = \begin{cases} +1, a_i^t = 1, \psi(\tilde{x}_i^t) \leq 0 | a_i^t = 0, \psi(\tilde{x}_i^t) \geq 0 \\ -1, a_i^t = 1, \psi(\tilde{x}_i^t) > 0 | a_i^t = 0, \psi(\tilde{x}_i^t) < 0 \end{cases} \tag{5}$$

where the positive exploratory reward plays a constructive guiding role in the conformal pseudo-label exploration, and vice versa. Then the target value $V_i^t$ can be defined as the sum of the instant exploratory reward $r_i^t$ and the future cumulative reward $Q(c_i^{t+1}, a_i^{t+1})$, formulated as follows:

$$V_i^t = r_i^t + \gamma Q(c_i^{t+1}, a_i^{t+1}) \tag{6}$$

**Algorithm 1** The PECL training process

---

**Input:** Training set $\mathcal{X} = \{X_i, O_i = \{x_i^j, y_i^j, b_i^j\}_{j=1}^{N_{il}}\}_{i=1}^{N}$
**Output:** Detector $\Theta$, conformal pseudo-label explorer $\pi$, and multi-clue selection evaluator $Q$
 1: Pre-train $\Theta$ with $X_i, O_i$ by minimizing $\mathcal{L}_{det}+\mathcal{L}_{pot}$, and construct the class-wise prototypes $\mathbb{P}$;
 2: Initialize the experience replay pool $D = \emptyset$;
 3: **for** $i = 1, 2, 3, ..., N$ **do**
 4:     Obtain the candidate set $\{\tilde{x}_i^t\}_{t=1}^{N_{ic}}$ for $X_i$ with $\Theta$;
 5:     Initialize the selected pseudo-label set $\tilde{O}_i^1 = \emptyset$ and the current candidate $\tilde{x}_i^1$, and then obtain the characteristic $c_i^1$;
 6:     **for** $t = 1, 2, 3, ..., N_{ic}$ **do**
 7:         # Perform the conformal pseudo-label explorer
 8:         Obtain the decision $a_i^t$ using $\pi$ with the current characteristic $c_i^t$;
 9:         Update $\tilde{O}_i^{t+1}$ by taking the decision $a_i^t$;
10:         # Conduct the multi-clue selection evaluator
11:         Obtain the next characteristic $c_i^{t+1}$ with $\tilde{x}_i^{t+1}$, $\tilde{O}_i^{t+1}$ and $O_i$;
12:         Calculate the reward $r_i^t$ by Eq. (5);
13:         Insert the recording $(c_i^t, a_i^t, r_i^t, c_i^{t+1})$ into $D$;
14:         Sample a batch from $D$ to update $\pi$ and $Q$;
15:     **end for**
16:     Update $\Theta$ with annotated objects $O_i$ and the selected pseudo-labels $\tilde{O}_i$ by minimizing $\mathcal{L}_{det}$ (Eq.(8)).
17: **end for**

---

where $\gamma$ is a discount factor. Finally, we can continuously update the conformal pseudo-label explorer $\pi$ and the multi-clue selection evaluator $Q$ through performing gradient descent on the following objective function:

$$\mathcal{L}_{ref} = \underbrace{(V_i^t - Q(c_i^t, a_i^t))^2}_{\text{Appro. target value}} - \underbrace{Q(c_i^t, a_i^t)}_{\text{Max. cum. reward}} \tag{7}$$

where the decision $a_i^t$ is sampled from $\pi(c_i^t)$. The first term aims to adjust the predicted cumulative reward to approximate the target value for updating the multi-clue selection evaluator $Q$. The second term aspires to maximize the cumulative reward, enabling the conformal pseudo-label explorer $\pi$ to refine the exploration policy. These selected high-quality candidates can in turn promote the capability of the aerial object detector. Furthermore, we use an experience replay pool $D$ to store a series of data $\{(c_i^t, a_i^t, r_i^t, c_i^{t+1})\}$ and sample batches to gradually update the parameters of the conformal pseudo-label explorer and the multi-clue selection evaluator (i.e., $\theta_{exp}$ and $\theta_{eva}$).

### 3.2.3 Progressive detector updating

The ultimate goal of our progressive exploration-conformable learning framework is to optimize an aerial detector by utilizing the sparse annotation and adaptively exploring these high-quality pseudo-labels in the training set. By employing these explored pseudo-labels $\tilde{O}_i^{N_{ic}}$ and the initial sparsely annotated objects $O_i$ as supervision signals, we can progressively acquire a satisfactory detector $\Theta$ by minimizing the loss function using stochastic gradient descent. Formally,

$$
\begin{aligned}
\mathcal{L}_{det} = &\sum_i^N \left(\frac{1}{|O_i|} \sum_{j=1}^{|O_i|} (\mathcal{L}_{cls}(x_i^j, y_i^j) + \mathcal{L}_{reg}(x_i^j, b_i^j) + \mathcal{L}_{pot}(x_i^j, \mathbb{P}))\right) \\
&+ \sum_i^N \left(\frac{1}{|\tilde{O}_i^{N_{ic}}|} \sum_{u=1}^{|\tilde{O}_i^{N_{ic}}|} (\mathcal{L}_{cls}(\tilde{x}_i^u, \tilde{y}_i^u) + \mathcal{L}_{reg}(\tilde{x}_i^u, \tilde{b}_i^u))\right)
\end{aligned}
\tag{8}
$$

where $|O_i|$ and $|\tilde{O}_i^{N_{ic}}|$ represent the number of sparsely annotated objects and the selected pseudo-labels during the conformal pseudo-label exploration process in the $i$-th image. $\tilde{y}_i^u$ and $\tilde{b}_i^u$ indicate the class label and bounding-box of the selected pseudo-labels $\tilde{x}_i^u$. $\mathcal{L}_{cls}$ is the standard cross-entropy loss used in ReDet (6) and OR-CNN (29) or the focal loss used in S$^2$A-Net (5) for classification, $\mathcal{L}_{reg}$ is the SmoothL1 loss for localization. To construct representative class-wise prototypes for providing confident guidance, we adopt the prototype loss $\mathcal{L}_{pot}$ as referenced in (38). To mitigate the impact of false negatives, we additionally employ the background weighting reduction strategy during the detector optimization process. Both the detector updating and the reinforced pseudo-label exploration are integrated to address the sparsely annotated aerial object detection task in a closed-loop, mutually reinforcing fashion. The PECL training process is presented in Algorithm 1.

Table 1: Performance comparisons of different detector baselines for the OBB task on the DOTA dataset at different label rates.

| | Method | PL | BD | BR | GTF | SV | LV | SH | TC | BC | ST | SBF | RA | HA | SP | HC | mAP(%) |
|---|---|---|---|---|---|---|---|---|---|---|---|---|---|---|---|---|---|
| **1%** | S²A-Net (5) | 57.27 | 50.85 | 20.13 | 59.11 | 36.98 | 32.58 | 37.97 | 83.75 | 49.15 | 27.89 | 42.48 | 48.70 | 18.33 | 24.54 | 12.35 | 40.14 |
| | S²A-Net w/ PECL | 73.30 | 63.17 | 25.57 | 62.35 | 55.87 | 48.41 | 59.71 | 90.00 | 66.99 | 53.07 | 37.35 | 50.43 | 27.13 | 38.45 | 12.18 | **50.39** |
| | OR-CNN (29) | 69.05 | 69.69 | 23.81 | 62.53 | 40.11 | 37.65 | 38.22 | 88.14 | 67.52 | 46.01 | 51.83 | 60.68 | 28.80 | 36.97 | 19.39 | 49.36 |
| | OR-CNN w/ PECL | 79.57 | 74.21 | 32.73 | 63.70 | 50.93 | 48.04 | 48.22 | 90.15 | 77.78 | 56.75 | 55.75 | 59.50 | 41.62 | 48.11 | 26.51 | **56.91** |
| | ReDet (6) | 60.36 | 62.98 | 26.16 | 66.08 | 34.24 | 39.99 | 29.35 | 87.42 | 68.58 | 50.11 | 49.94 | 56.59 | 30.75 | 40.43 | 18.51 | 48.10 |
| | ReDet w/ PECL | 86.09 | 72.16 | 38.91 | 65.05 | 67.29 | 72.01 | 75.40 | 89.17 | 78.52 | 70.42 | 56.75 | 57.68 | 54.09 | 53.50 | 18.72 | **63.72** |
| **2%** | S²A-Net (5) | 56.92 | 48.39 | 21.41 | 61.37 | 42.76 | 35.06 | 46.46 | 85.02 | 54.32 | 35.45 | 38.11 | 50.41 | 20.56 | 29.15 | 3.35 | 41.92 |
| | S²A-Net w/ PECL | 74.57 | 58.63 | 26.14 | 62.36 | 60.68 | 55.81 | 69.71 | 89.64 | 65.52 | 62.58 | 36.53 | 52.20 | 31.45 | 42.58 | 18.71 | **53.81** |
| | OR-CNN (29) | 68.78 | 71.00 | 24.81 | 63.66 | 47.06 | 37.65 | 45.59 | 87.38 | 70.69 | 51.26 | 53.54 | 57.26 | 28.20 | 44.52 | 23.40 | 51.65 |
| | OR-CNN w/ PECL | 79.80 | 72.87 | 31.12 | 67.52 | 58.97 | 51.18 | 66.16 | 90.15 | 74.48 | 58.96 | 53.41 | 60.98 | 42.97 | 44.95 | 26.43 | **58.66** |
| | ReDet (6) | 59.73 | 64.67 | 23.65 | 72.47 | 40.07 | 40.37 | 53.99 | 87.90 | 64.66 | 48.02 | 51.16 | 57.63 | 33.20 | 46.65 | 23.79 | 51.20 |
| | ReDet w/ PECL | 87.27 | 73.30 | 41.54 | 70.35 | 67.19 | 71.88 | 81.14 | 89.08 | 73.33 | 77.97 | 55.47 | 61.82 | 60.77 | 55.55 | 32.09 | **66.58** |
| **5%** | S²A-Net (5) | 65.85 | 49.72 | 21.97 | 59.97 | 56.63 | 52.17 | 66.53 | 87.14 | 57.84 | 48.72 | 38.07 | 50.47 | 20.78 | 36.08 | 19.75 | 48.78 |
| | S²A-Net w/ PECL | 78.18 | 63.18 | 30.36 | 63.85 | 66.22 | 64.50 | 73.06 | 90.26 | 69.54 | 64.57 | 41.92 | 50.56 | 32.99 | 43.54 | 28.56 | **57.42** |
| | OR-CNN (29) | 78.13 | 71.05 | 24.62 | 64.69 | 57.81 | 57.64 | 70.99 | 88.62 | 67.80 | 53.27 | 49.46 | 60.68 | 30.71 | 37.51 | 31.46 | 56.30 |
| | OR-CNN w/ PECL | 86.95 | 73.10 | 30.17 | 67.68 | 68.34 | 69.61 | 80.10 | 90.07 | 74.91 | 71.95 | 47.80 | 59.82 | 48.26 | 52.43 | 40.06 | **64.08** |
| | ReDet (6) | 78.21 | 68.73 | 31.49 | 69.65 | 54.25 | 57.11 | 63.27 | 88.38 | 62.21 | 49.87 | 52.09 | 56.34 | 41.12 | 44.69 | 23.52 | 56.06 |
| | ReDet w/ PECL | 88.38 | 71.19 | 36.56 | 64.78 | 72.50 | 71.38 | 82.51 | 89.65 | 75.85 | 76.66 | 50.79 | 60.95 | 61.53 | 61.00 | 42.08 | **67.06** |
| **10%** | S²A-Net (5) | 73.92 | 57.80 | 28.38 | 62.52 | 63.08 | 64.10 | 71.33 | 88.10 | 59.29 | 60.02 | 42.47 | 52.71 | 33.22 | 44.55 | 14.21 | 54.39 |
| | S²A-Net w/ PECL | 80.40 | 75.51 | 38.31 | 65.62 | 65.93 | 70.60 | 69.21 | 88.97 | 78.77 | 69.69 | 53.13 | 56.31 | 52.84 | 53.29 | 18.71 | **62.49** |
| | OR-CNN (29) | 77.51 | 69.97 | 33.06 | 63.23 | 68.25 | 69.09 | 75.27 | 88.36 | 65.87 | 63.19 | 52.26 | 65.89 | 42.29 | 45.69 | 51.72 | 62.11 |
| | OR-CNN w/ PECL | 87.07 | 75.11 | 37.59 | 65.78 | 69.88 | 72.53 | 82.44 | 90.47 | 77.33 | 75.82 | 53.24 | 65.07 | 54.58 | 59.12 | 50.11 | **67.74** |
| | ReDet (6) | 78.75 | 70.17 | 32.21 | 69.76 | 59.90 | 67.18 | 73.86 | 87.06 | 66.62 | 58.84 | 53.44 | 56.45 | 50.91 | 43.70 | 35.89 | 60.32 |
| | ReDet w/ PECL | 87.23 | 75.15 | 42.24 | 67.21 | 74.05 | 73.95 | 85.42 | 90.41 | 84.47 | 71.95 | 59.35 | 57.83 | 64.48 | 61.36 | 45.30 | **69.36** |

# 4 Experiments

## 4.1 Dataset and setup

**Datasets:** We conduct experiments on two public aerial datasets, DOTA (27) and HRSC2016 (15), to evaluate the performance of our proposed PECL. The DOTA (27) dataset consists of 2806 high-resolution aerial images, where the fully annotated 188282 instances vary in size, aspect ratio and rotation angle. The dataset has 15 object categories, including plane (PL), baseball-diamond (BD), bridge (BR), ground-track-field (GTF), small-vehicle (SV), large-vehicle (LV), ship (SH), soccer-ball-field (SBF), tennis-court (TC), basketball-court (BC), storage-tank (ST), roundabout (RA), harbor (HA), swimming-pool (SP), and helicopter (HC). To meet the requirement of sparse annotation, we adopt a novel protocol where we randomly sample 1%, 2%, 5% and 10% of instances for each class for annotation. While the HRSC2016 (15) dataset contains aerial images with complex backgrounds and diverse ship objects. The training set and validation set contain 436 and 181 images, while the test set consists of 444 images. Here 1% and 2% label rates often lead to over-fitting due to the HRSC2016 dataset is with a small amount of images, thus we only generate sparsely labeled versions of the dataset with 5% and 10% label rates. All evaluation experiments are conducted on the test set with the standard mean average precision (mAP) as the performance evaluation metric.

**Implementation details:** To demonstrate the generality of our PECL, we adopt two-stage detectors, including ReDet (6) and OR-CNN (29), as well as a one-stage detector, S²A-Net (5), serving as the baselines. In our experiments, we set the following hyper-parameters: $K$=10, $\xi$=1, $\gamma$=0.9. The feature dimension $d$ is set to 1024 for ReDet and OR-CNN, 256 for S²A-Net. Our models are trained with the mmdetection (4)/mmrotate (39) framework on the DOTA dataset for 12 epochs, and on the HRSC2016 dataset for 36 epochs. All the experiments are conducted on two NVIDIA 2080Ti GPUs. For performing the detector optimization, we employ the SGD optimizer with an initial learning rate of 0.01. The learning rate is divided by 10 at the decay steps: 8 and 11 for DOTA, 24 and 33 for HRSC2016. The momentum and weight decay are set to 0.9 and 1e-4. The background/foreground weights are set to 0.3 and 1.0, respectively. The conformal pseudo-label explorer and multi-clue selection evaluator adopt the SGD optimization with learning rates of 1e-3 and 5e-4, respectively. The maximum capacity of an experience replay pool is set to 1000.

## 4.2 Experimental results

**DOTA:** We present the detection results for the oriented bounding box (OBB) task and horizontal bounding box (HBB) task on the DOTA (27) dataset, as shown in Table 1 and Table 2, respectively. In general, our PECL has demonstrated remarkable improvements over different baselines across all

Table 2: Performance comparisons of different detector baselines for the HBB task on the DOTA dataset at different label rates.

| Method | mAP(%) | | | |
|---|---|---|---|---|
| | 1% | 2% | 5% | 10% |
| S$^2$A-Net (5) | 40.04 | 42.07 | 48.84 | 53.00 |
| S$^2$A-Net w/ PECL | **50.67** | **53.28** | **57.82** | **62.57** |
| OR-CNN (29) | 49.31 | 52.91 | 57.26 | 63.12 |
| OR-CNN w/ PECL | **57.58** | **57.93** | **63.74** | **67.55** |
| ReDet (6) | 48.57 | 52.69 | 57.93 | 61.22 |
| ReDet w/ PECL | **64.70** | **67.46** | **67.76** | **69.95** |

Table 3: Performance comparisons of different detector baselines for the OBB and HBB task on the HRSC2016 at different label rates.

| Method | mAP(%) | | | |
|---|---|---|---|---|
| | 5% | | 10% | |
| | OBB | HBB | OBB | HBB |
| S$^2$A-Net (5) | 57.90 | 65.09 | 68.42 | 79.23 |
| S$^2$A-Net w/PECL | **75.50** | **75.84** | **80.65** | **85.88** |
| OR-CNN (29) | 40.90 | 52.48 | 55.70 | 63.22 |
| OR-CNN w/PECL | **54.30** | **62.46** | **85.50** | **85.68** |
| ReDet (6) | 58.28 | 69.36 | 79.66 | 83.36 |
| ReDet w/PECL | **75.10** | **80.42** | **87.29** | **88.57** |

Table 4: Comparison with state-of-the-art methods for the OBB task on the DOTA dataset at the 5% label rate. The detection methods are based on rotated-Faster-RCNN ($*$) and rotated-RetinaNet ($\dagger$).

| Setting | Method | mAP(%) |
|---|---|---|
| Supervised | S$^2$A-Net$^\dagger$ (5) | 48.78 |
| | ReDet$^*$ (6) | 56.06 |
| Semi-supervised | SOOD$^\dagger$ (7) | 53.74 |
| | Unbiased Teacher$^*$ (14) | 64.74 |
| Sparse-annotated | Calibrated Teacher$^\dagger$ (25) | 55.81 |
| | **S$^2$A-Net$^\dagger$ w/PECL** | **57.42** |
| | BRL$^*$ (35) | 65.04 |
| | Co-mining$^*$ (26) | 65.35 |
| | Region-based$^*$ (19) | 65.71 |
| | **ReDet$^*$ w/PECL** | **67.06** |

Table 5: Performance comparisons of different strategies when selecting pseudo-labels on the DOTA dataset at the 1% label rate.

| Setting | Pseudo-label explorer | Selection evaluator | Experience replay mechanism | mAP(%) |
|---|---|---|---|---|
| I | - | - | - | 48.10 |
| II | ✓ | - | - | 60.84 |
| III | ✓ | ✓ | - | 61.31 |
| IV | ✓ | ✓ | ✓ | **63.72** |

Table 6: Performance comparisons of different exploratory characteristic designs on the DOTA dataset at the 1% label rate.

| Setting | Predicted probability | Feature similarity | Confidence level | mAP(%) |
|---|---|---|---|---|
| I | ✓ | ✓ | - | 60.90 |
| II | ✓ | - | ✓ | 61.17 |
| III | - | ✓ | ✓ | 62.40 |
| IV | ✓ | ✓ | ✓ | **63.72** |

label rates. For the OBB task, PECL achieves performance gains of 10.25%, 7.55%, and 15.62% at the 1% label rate compared with the baselines, indicating the effectiveness of our approach on different detectors. Furthermore, it is worth noting that our PECL shows greater performance improvement in some densely distributed small objects, such as small-vehicle (SV) and ship (SH), compared to individually presented large objects, such as baseball-diamond (BD) and soccer-ball-field (SBF) (33.05%, 46.05% *vs* 9.18%, 6.81% at the 1% label rate with ReDet (6)). This demonstrates that our PECL takes full advantage of the contextual relationships between instances.

Similarly, for the HBB task, our PECL boosts the performance of the detectors at all label rates, achieving at least 4.43% improvement. Importantly, as to the ReDet baseline, our sparsely trained model could achieve a competitive performance to the fully supervised one (69.95% vs 77.47%) by only using a label rate of 10%.

**HRSC2016:** We further conduct comparative experiments on the HRSC2016 dataset (15), and the performance comparisons of our PECL with different detector baselines for two tasks are presented in Table 3. Overall, our method has brought great improvements across various experimental settings. At the 10% label rate, our PECL achieves mAPs of 80.65%, 85.50% and 87.29% for the OBB task, surpassing the baselines with gains of 12.23%, 29.80% and 7.63%, respectively. Similar improvements are observed at the 5% label rate. These results underscore the effectiveness of our PECL even on relatively small-scale aerial dataset, further highlighting its efficacy in sparsely annotated aerial object detection task.

To demonstrate the superiority of our PECL, we further compare it with other state-of-the-art semi/sparse-supervised methods applied to the sparsely annotated aerial object detection task. To ensure fairness, all comparative experiments are conducted with ResNet50/ReR50-ReFPN on the DOTA dataset at the 5% label rate. The results are presented in Table 4. It can be observed that our PECL achieves the best performance of 67.06% with ReDet as the baseline, surpassing Unbiased Teacher, BRL, Co-mining, and Region-based by 2.32%, 2.02%, 1.71%, and 1.35%, respectively. When S$^2$A-Net is used as the baseline, our PECL is 3.68% and 1.61% higher in mAP compared

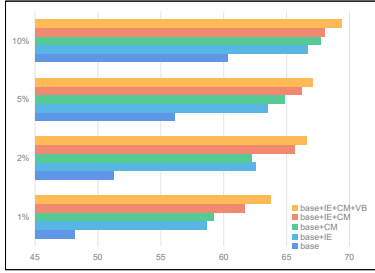
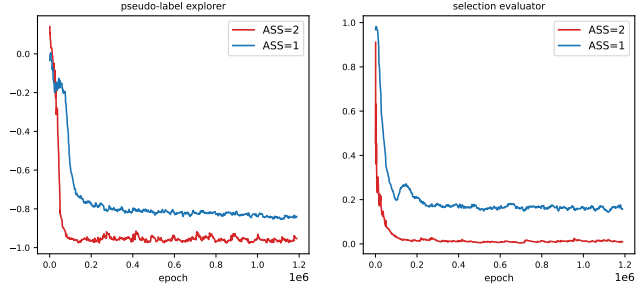

Figure 2: Performance comparisons of different reward settings at various label rates on the DOTA dataset.

Figure 3: Loss curves of conformal pseudo-label explorer and multi-clue selection evaluator under different action spaces at the 1% label rate on the DOTA dataset.

to SOOD and Calibrated Teacher. The reason may be that our approach can explore high-quality pseudo-labels by addressing the imbalanced prediction probabilities across different categories in the aerial scenes, thereby mitigating the limitation of sparse annotation and boosting the detector optimization.

### 4.3 Ablation study

We conduct comprehensive ablation experiments to evaluate the performance of our framework under different settings. All the experiments are conducted with the ReDet baseline on the DOTA dataset at the 1% label rate for the OBB task, unless stated otherwise.

**Effect of each component:** As shown in Table 5, we give comparative results with three strategies when generating an optimal pseudo-label selection policy, including the conformal pseudo-label explorer, the multi-clue selection evaluator, and the experience replay mechanism. When only using the conformal pseudo-label explorer, we update the policy using the policy gradient method (22) and the detector achieves an mAP of 60.84%. It already outperforms the baseline, verifying the effectiveness of the conformal pseudo-label exploration mechanism. When further adopting the multi-clue selection evaluator, we can achieve more improvement with an mAP of 61.31%. The reason may be that the evaluator provides positive guidance for policy optimization. Moreover, the accumulated experience can alleviate overestimation and bring the best mAP of 63.72%.

**Reward setting in the multi-clue selection evaluator:** We study the effects of three factors of the designed instant exploratory reward, including the information entropy (IE), confidence margin (CM) and value binarization (VB) technique in Eq. (5). As presented in Figure 2, the detection results can be improved with different factors of the exploratory reward to some extent. These results indicate that combining information entropy and confidence margin as the exploratory reward is the most effective way to measure the uncertainty/quality of candidates and evaluate the pseudo-label selection. Besides, the VB technique ensures the stable convergence for policy optimization, further enhancing the detection performance.

**Exploratory characteristic design in the conformal pseudo-label explorer:** To validate the significance of different characteristics when forming the exploratory characteristic, Table 6 presents its comparative results with the predicted probability, feature similarity and confidence level. The predicted probability can measure the accuracy of the candidate, resulting in 1.32% performance boost. The feature similarity indicates the resemblance of the candidate to the class centers, leading to a gain of 2.55%. The confidence level reflects the inconsistency between the candidate and the corresponding prototypes, contributing to 2.82% improvement. These results confirm the rationality of our characteristic design and emphasize the importance of appropriate exploratory characteristic to enhance the performance.

**Action space in the conformal pseudo-label explorer:** We design two sizes of action space (ASS), namely the dimension of the probability vector obtained by the pseudo-label explorer. For 'ASS=1', the explorer outputs the probability of selecting the candidate, while 'ASS=2' gives the likelihood of the candidate being selected or not. Our approach achieves performance of 60.81% ('ASS=1') *vs* 63.72% ('ASS=2') on the DOTA dataset at the 1% label rate. Moreover, Figure 3 illustrates the loss curves of explorer and evaluator under different action spaces. We can observe when 'ASS=2', the loss values converge faster and reach lower convergence values compared to 'ASS=1'. The

reason may be that 'ASS=2' could provide a larger decision space, thereby promoting more flexible decision-making and improving the exploration capability of the pseudo-label explorer.

## 5 Conclusion and Discussion

In this paper, we propose a Progressive Exploration-Conformable Learning (PECL) framework to address the sparsely annotated aerial object detection problem, which can adaptively perform the conformal pseudo-label exploration from the large-scale unlabeled aerial images. By introducing the conformal pseudo-label explorer and the multi-clue selection evaluator, we formulate the pseudo-label exploration as a conformal decision-making problem. It can select these confident pseudo-labels by considering the specific characteristics of different categories and inter-instance contextual relationships. Extensive experiments and ablation studies demonstrate the effectiveness of our PECL framework.

**Limitations.** This work also has some limitations. The first limitation is that our work is built for scenes with densely distributed objects, and its performance in general scenes may be suboptimal. Secondly, the application of reinforcement learning algorithms increases the overall training time. In the future, we will continue to explore the potential of sparsely annotated object detection and extend this PECL framework to other weakly supervised detection/segmentation tasks in general/aerial images to further reduce the annotation costs.

## 6 Acknowledgement

This work is supported by the National Natural Science Foundation of China (Grants Nos. 62372238,62476133), the Natural Science Foundation of Shandong Province China (Grants Nos. ZR2022LZH003, ZR2020LZH008).

## References

[1] Walid Abdullah Al and Il Dong Yun. Partial policy-based reinforcement learning for anatomical landmark localization in 3d medical images. *IEEE Transactions on Medical Imaging*, 39(4):1245–1255, 2019.

[2] David Berthelot, Nicholas Carlini, Ian Goodfellow, Nicolas Papernot, Avital Oliver, and Colin A Raffel. Mixmatch: A holistic approach to semi-supervised learning. *Advances in Neural Information Processing Systems*, 32, 2019.

[3] Juan C Caicedo and Svetlana Lazebnik. Active object localization with deep reinforcement learning. In *Proceedings of the IEEE International Conference on Computer Vision*, pages 2488–2496, 2015.

[4] Kai Chen, Jiaqi Wang, Jiangmiao Pang, Yuhang Cao, Yu Xiong, Xiaoxiao Li, Shuyang Sun, Wansen Feng, Ziwei Liu, Jiarui Xu, et al. Mmdetection: Open mmlab detection toolbox and benchmark. *arXiv preprint arXiv:1906.07155*, 2019.

[5] Jiaming Han, Jian Ding, Jie Li, and Gui-Song Xia. Align deep features for oriented object detection. *IEEE Transactions on Geoscience and Remote Sensing*, 60:1–11, 2021.

[6] Jiaming Han, Jian Ding, Nan Xue, and Gui-Song Xia. Redet: A rotation-equivariant detector for aerial object detection. In *Proceedings of the IEEE Conference on Computer Vision and Pattern Recognition*, pages 2786–2795, 2021.

[7] Wei Hua, Dingkang Liang, Jingyu Li, Xiaolong Liu, Zhikang Zou, Xiaoqing Ye, and Xiang Bai. Sood: Towards semi-supervised oriented object detection. In *Proceedings of the IEEE Conference on Computer Vision and Pattern Recognition*, pages 15558–15567, 2023.

[8] Chen Huang, Simon Lucey, and Deva Ramanan. Learning policies for adaptive tracking with deep feature cascades. In *Proceedings of the IEEE International Conference on Computer Vision*, pages 105–114, 2017.

[9] Jisoo Jeong, Seungeui Lee, Jeesoo Kim, and Nojun Kwak. Consistency-based semi-supervised learning for object detection. *Advances in Neural Information Processing Systems*, 32, 2019.

[10] Dong-Hyun Lee. Pseudo-label: The simple and efficient semi-supervised learning method for deep neural networks. In *Proceedings of International Conference on Machine Learning*, volume 3, page 896, 2013.

[11] Junnan Li, Caiming Xiong, and Steven CH Hoi. Comatch: Semi-supervised learning with contrastive graph regularization. In *Proceedings of the IEEE International Conference on Computer Vision*, pages 9475–9484, 2021.

[12] Tsung-Yi Lin, Priya Goyal, Ross Girshick, Kaiming He, and Piotr Dollár. Focal loss for dense object detection. In *Proceedings of the IEEE International Conference on Computer Vision*, pages 2980–2988, 2017.

[13] Tsung-Yi Lin, Michael Maire, Serge Belongie, James Hays, Pietro Perona, Deva Ramanan, Piotr Dollár, and C Lawrence Zitnick. Microsoft coco: Common objects in context. In *Proceedings of European Conference on Computer Vision*, pages 740–755, 2014.

[14] Yen-Cheng Liu, Chih-Yao Ma, Zijian He, Chia-Wen Kuo, Kan Chen, Peizhao Zhang, Bichen Wu, Zsolt Kira, and Peter Vajda. Unbiased teacher for semi-supervised object detection. In *Proceedings of International Conference on Learning Representations*, 2021.

[15] Zikun Liu, Liu Yuan, Lubin Weng, and Yiping Yang. A high resolution optical satellite image dataset for ship recognition and some new baselines. In *International Conference on Pattern Recognition Applications and Methods*, volume 2, pages 324–331, 2017.

[16] Junwei Luo, Xue Yang, Yi Yu, Qingyun Li, Junchi Yan, and Yansheng Li. Pointobb: Learning oriented object detection via single point supervision. In *Proceedings of the IEEE Conference on Computer Vision and Pattern Recognition*, pages 16730–16740, 2024.

[17] Romany F Mansour, José Escorcia-Gutierrez, Margarita Gamarra, Jair A Villanueva, and Nallig Leal. Intelligent video anomaly detection and classification using faster rcnn with deep reinforcement learning model. *Image and Vision Computing*, 112:104229, 2021.

[18] Yusuke Niitani, Takuya Akiba, Tommi Kerola, Toru Ogawa, Shotaro Sano, and Shuji Suzuki. Sampling techniques for large-scale object detection from sparsely annotated objects. In *Proceedings of the IEEE Conference on Computer Vision and Pattern Recognition*, pages 6510–6518, 2019.

[19] Sai Saketh Rambhatla, Saksham Suri, Rama Chellappa, and Abhinav Shrivastava. Sparsely annotated object detection: A region-based semi-supervised approach. In *Proceedings of the IEEE International Conference on Computer Vision*, 2023.

[20] Glenn Shafer and Vladimir Vovk. A tutorial on conformal prediction. *Journal of Machine Learning Research*, 9(3), 2008.

[21] Kihyuk Sohn, Zizhao Zhang, Chun-Liang Li, Han Zhang, Chen-Yu Lee, and Tomas Pfister. A simple semi-supervised learning framework for object detection. *arXiv preprint arXiv:2005.04757*, 2020.

[22] Richard S Sutton, David McAllester, Satinder Singh, and Yishay Mansour. Policy gradient methods for reinforcement learning with function approximation. *Advances in Neural Information Processing Systems*, 12, 1999.

[23] Yihe Tang, Weifeng Chen, Yijun Luo, and Yuting Zhang. Humble teachers teach better students for semi-supervised object detection. In *Proceedings of the IEEE Conference on Computer Vision and Pattern Recognition*, pages 3132–3141, 2021.

[24] Zhiqiang Tian, Xiangyu Si, Yaoyue Zheng, Zhang Chen, and Xiaojian Li. Multi-step medical image segmentation based on reinforcement learning. *Journal of Ambient Intelligence and Humanized Computing*, pages 1–12, 2020.

[25] Haohan Wang, Liang Liu, Boshen Zhang, Jiangning Zhang, Wuhao Zhang, Zhenye Gan, Yabiao Wang, Chengjie Wang, and Haoqian Wang. Calibrated teacher for sparsely annotated object detection. In *Proceedings of the AAAI Conference on Artificial Intelligence*, 2023.

[26] Tiancai Wang, Tong Yang, Jiale Cao, and Xiangyu Zhang. Co-mining: Self-supervised learning for sparsely annotated object detection. In *Proceedings of the AAAI Conference on Artificial Intelligence*, volume 35, pages 2800–2808, 2021.

[27] Gui-Song Xia, Xiang Bai, Jian Ding, Zhen Zhu, Serge Belongie, Jiebo Luo, Mihai Datcu, Marcello Pelillo, and Liangpei Zhang. Dota: A large-scale dataset for object detection in aerial images. In *Proceedings of the IEEE Conference on Computer Vision and Pattern Recognition*, pages 3974–3983, 2018.

[28] Qizhe Xie, Zihang Dai, Eduard Hovy, Thang Luong, and Quoc Le. Unsupervised data augmentation for consistency training. *Advances in Neural Information Processing Systems*, 33:6256–6268, 2020.

[29] Xingxing Xie, Gong Cheng, Jiabao Wang, Xiwen Yao, and Junwei Han. Oriented r-cnn for object detection. In *Proceedings of the IEEE International Conference on Computer Vision*, pages 3520–3529, 2021.

[30] Mengde Xu, Zheng Zhang, Han Hu, Jianfeng Wang, Lijuan Wang, Fangyun Wei, Xiang Bai, and Zicheng Liu. End-to-end semi-supervised object detection with soft teacher. In *Proceedings of the IEEE International Conference on Computer Vision*, pages 3060–3069, 2021.

[31] Xue Yang, Gefan Zhang, Wentong Li, Xuehui Wang, Yue Zhou, and Junchi Yan. H2rbox: Horizontal box annotation is all you need for oriented object detection. *arXiv preprint arXiv:2210.06742*, 2022.

[32] Jihun Yoon, Seungbum Hong, and Min-Kook Choi. Semi-supervised object detection with sparsely annotated dataset. In *Proceedings of the IEEE International Conference on Image Processing*, pages 719–723, 2021.

[33] Ke Yu, Chao Dong, Liang Lin, and Chen Change Loy. Crafting a toolchain for image restoration by deep reinforcement learning. In *Proceedings of the IEEE Conference on Computer Vision and Pattern Recognition*, pages 2443–2452, 2018.

[34] Yi Yu, Xue Yang, Qingyun Li, Feipeng Da, Jifeng Dai, Yu Qiao, and Junchi Yan. Point2rbox: Combine knowledge from synthetic visual patterns for end-to-end oriented object detection with single point supervision. In *Proceedings of the IEEE Conference on Computer Vision and Pattern Recognition*, pages 16783–16793, 2024.

[35] Han Zhang, Fangyi Chen, Zhiqiang Shen, Qiqi Hao, Chenchen Zhu, and Marios Savvides. Solving missing-annotation object detection with background recalibration loss. In *Proceedings of the IEEE International Conference on Acoustics, Speech and Signal Processing*, pages 1888–1892, 2020.

[36] Hongyu Zhou, Zheng Ge, Songtao Liu, Weixin Mao, Zeming Li, Haiyan Yu, and Jian Sun. Dense teacher: Dense pseudo-labels for semi-supervised object detection. In *Proceedings of European Conference on Computer Vision*, pages 35–50, 2022.

[37] Kaiyang Zhou, Yu Qiao, and Tao Xiang. Deep reinforcement learning for unsupervised video summarization with diversity-representativeness reward. In *Proceedings of the AAAI Conference on Artificial Intelligence*, volume 32, 2018.

[38] Tianfei Zhou, Wenguan Wang, Ender Konukoglu, and Luc Van Gool. Rethinking semantic segmentation: A prototype view. In *Proceedings of the IEEE Conference on Computer Vision and Pattern Recognition*, pages 2582–2593, 2022.

[39] Yue Zhou, Xue Yang, Gefan Zhang, Jiabao Wang, Yanyi Liu, Liping Hou, Xue Jiang, Xingzhao Liu, Junchi Yan, Chengqi Lyu, et al. Mmrotate: A rotated object detection benchmark using pytorch. In *Proceedings of the 30th ACM International Conference on Multimedia*, pages 7331–7334, 2022.

## A  Advancing together with big foundation models.

To adapt to the big model era, we adopt big-model CLIP for sparsely annotated object detection task. The comparisons are shown in Table 7. We observe that direct use of CLIP for pseudo-label inference cannot cause a large improvement, but additionally adding our PECL boosts the performance greatly. A possible reason is the difference between aerial images and general images (trained for CLIP).

Table 7: Comparisons after adopting the CLIP model.

| Method | 1% | 2% | 5% | 10% |
|---|---|---|---|---|
| ReDet | 48.10 | 51.20 | 56.06 | 60.32 |
| ReDet w/ CLIP | 54.01 | 56.33 | 60.87 | 62.27 |
| ReDet w/ PECL | 63.72 | 66.58 | 67.06 | 69.36 |
| ReDet w/ PECL+CLIP | **64.75** | **67.65** | **68.36** | **69.84** |

## B  Comparison with SOTA oriented object detection methods.

We have compared with some SOTA oriented object detection methods (e.g., $S^2$A-Net, OR-CNN and ReDet) in sparsely annotated task, as in *Tables 1, 2, 3 of the submitted paper*. Here, we add a comparison with the latest oriented method LSKNet-T on the DOTA dataset at the 5% label rate, as shown in Table 8. It indicates that our PECL can be well-generalized to SOTA oriented detectors.

Table 8: Performance comparison with SOTA oriented detector as baselines at the 5% label rate on the DOTA dataset.

| Method | LSKNet-T | LSKNet-T w/ PECL |
|---|---|---|
| mAP(%) | 58.08 | **67.89** |

## C  Performance comparisons with more label rates setting.

We follow the protocol of previous semi-supervised method (e.g., STAC) to utilize the extreme label rates of 1%, 2%, 5%, 10%, which actually consider the abundance of objects in aerial images. For the label rates, e.g., 30%, 50%, 70%, 100%, we also report results on the DOTA dataset in Table 9. It can be observed that our PECL can bring certain performance improvement under all sparse label rates, indicating the effectiveness of our method in the sparsely annotated aerial object detection tasks. However, under the 100% label rate, the performance deteriorates, possibly because the method introduces other label noise under full supervision, which affects the performance of the detector.

Table 9: Performance comparisons with more label rates.

| Method | 30% | 50% | 70% | 100% |
|---|---|---|---|---|
| ReDet | 67.48 | 71.44 | 72.46 | 76.25 |
| ReDet w/ PECL | **73.03** | **73.95** | **74.05** | **75.18** |

## D  Visualization results

The qualitative comparisons of detection results are visualized in Figure 4. We can observe that our proposed PECL outputs more meaningful and precise predictions than Region-based and the baseline ReDet on all different tasks and datasets, especially in some densely distributed small instances, i.e., plane/ship. But our method seems to be not precise enough in location, which is the future research. These visualization results demonstrate the superiority of our PECL in SAAOD task.

## E  Description of prototype loss

Here we describe in detail the prototype loss $L_{pot}$ in the *Eq.(8) of the submitted paper*. In our proposed PECL framework, we construct class-wise prototypes $\mathbb{P}$ to provide prior confident guidance

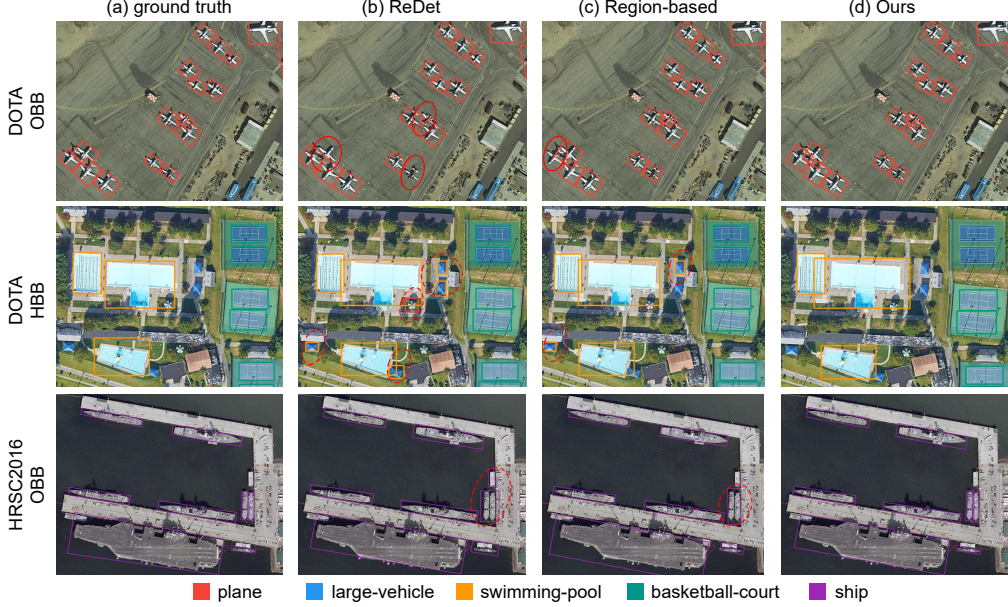

Figure 4: Some visualized detection results at the 10% label rate. From left to right, each column shows the results of the ground truth, supervised baseline (ReDet), Region-based, and our PECL. From top to bottom, each row represents the results of OBB task on DOTA dataset, HBB task on DOTA dataset, and OBB task on HRSC2016 dataset. The colored rectangles represent predictions. The red dashed circles and the solid red circles indicate false and missed detections, respectively.

for the subsequent reinforced pseudo-label exploration process. In order to obtain more robust and representative prototypes, the prototype loss consists of two components, expressed as:

$$\mathcal{L}_{pot} = \mathcal{L}_{cont} + \mathcal{L}_{ret} \tag{9}$$

where $\mathcal{L}_{cont}$ represents a contrastive learning loss between sparsely annotated instances and prototypes to distinguish prototypes of different categories, expressed as:

$$\mathcal{L}_{cont} = -\log \frac{\exp(\cos \theta_{y_i^j k_i^j})}{\sum_{c=1}^{C}\sum_{k=1}^{K} \exp\left(\cos \theta_{ck}\right)} \tag{10}$$

where $\cos \theta_{ck} = p_{ck} \cdot F(x_i^j)^T$ is the cosine similarity between the feature vector of instance $x_i^j$ and the $c$-th class $k$-th prototype, $F(\cdot)$ extracts the feature of the last fully connected layer. $k_i^j = \arg\max_k \{\cos \theta_{y_i^j k}\}_{k=1}^{K}$ represents the $k_i^j$-th prototype of class $y_i^j$. While $\mathcal{L}_{ret}$ is a regularization term to minimize the distance between the instances and the intra-class prototypes. The term can be formulated as follows:

$$\mathcal{L}_{ret} = (1 - p_{y_i^j k_i^j} \cdot F(x_i^j))^2 \tag{11}$$

By applying these constraints, we constantly evolve the prototypes through momentum update method during the detector optimization. Formally,

$$p_{ck} = \alpha p_{ck} + (1 - \alpha) F(x_i^j) \tag{12}$$

where $\alpha \in [0, 1]$ is a momentum coefficient.

## F  Ablation for prototype number

The number $K$ determines the number of class centers and the size of the calibration set used for confidence level, greatly affecting the subsequent reinforced pseudo-label exploration process. We present the performance comparisons of different prototype numbers on DOTA dataset at the 1% label rate in Table 10. It could be observed that when $K = 10$, the detector performance reaches the best

63.72% mAP. When $K$=5, the performance decreases, but it is still a considerable performance. When $K$ rises to 50, it brings a certain degree of drop. We believe that it may be due to the redundant class centers, or sparse annotation that cannot learn prototypes well. Therefore, $K$=10 can better represent the probability distribution of classes, calculate the confidence level of candidates, and construct better exploratory characteristic, bringing balance between representation and computational complexity.

Table 10: Performance comparisons with different numbers of prototypes for each class at the 1% label rate on the DOTA dataset.

| $K$ | 5 | 10 | 20 | 50 |
|---|---|---|---|---|
| mAP(%) | 61.77 | **63.72** | 62.91 | 63.18 |

